# Compressive Sensing MRI with Wavelet Tree Sparsity

**Chen Chen and Junzhou Huang**
Department of Computer Science and Engineering
University of Texas at Arlington
`cchen@mavs.uta.edu`
`jzhuang@uta.edu`

## Abstract

In Compressive Sensing Magnetic Resonance Imaging (CS-MRI), one can reconstruct a MR image with good quality from only a small number of measurements. This can significantly reduce MR scanning time. According to structured sparsity theory, the measurements can be further reduced to $\mathcal{O}(K + \log n)$ for tree-sparse data instead of $\mathcal{O}(K + K \log n)$ for standard $K$-sparse data with length $n$. However, few of existing algorithms have utilized this for CS-MRI, while most of them model the problem with total variation and wavelet sparse regularization. On the other side, some algorithms have been proposed for tree sparse regularization, but few of them have validated the benefit of wavelet tree structure in CS-MRI. In this paper, we propose a fast convex optimization algorithm to improve CS-MRI. Wavelet sparsity, gradient sparsity and tree sparsity are all considered in our model for real MR images. The original complex problem is decomposed into three simpler subproblems then each of the subproblems can be efficiently solved with an iterative scheme. Numerous experiments have been conducted and show that the proposed algorithm outperforms the state-of-the-art CS-MRI algorithms, and gain better reconstructions results on real MR images than general tree based solvers or algorithms.

## 1  Introduction

Magnetic Resonance Imaging (MRI) is widely used for observing the tissue changes of the patients within a non-invasive manner. One limitation of MRI is its imaging speed, including both scanning speed and reconstruction speed. Long waiting time and slow scanning may result in patients' annoyance and blur on images due to local motion such as breathing, heart beating etc. According to compressive sensing (CS) [1,2] theory, only a small number of measurements is enough to recover an image with good quality. This is an extension of Nyquist-Shannon sampling theorem when data is sparse or can be sparsely represented. Compressive Sensing Magnetic Resonance Imaging (CS-MRI) becomes one of the most successful applications of compressive sensing, since MR scanning time is directly related to the number of sampling measurements [3]. As most images can be transferred to some sparse domain (wavelet etc.), only $\mathcal{O}(K + K \log n)$ samples are enough to obtain robust MR image reconstruction.

Actually, this result can be better. Recent works on structured sparsity show that the required number of sampling measurements could be further reduced to $\mathcal{O}(K + \log n)$ by exploring the tree structure [4-6]. A typical relationship in tree sparsity is that, if a parent coefficient has a large/small value, its children also tend to be large/small. Some methods have been proposed to improve standard CS reconstruction by utilizing this prior. Specially, two convex models are proposed to handle the tree-based reconstruction problem [7]. They apply SpaRSA [11] to solve their models, with a relatively slow convergence rate. In Bayesian compressive sensing, Markov Chain Monte Carlo (MCMC) and variational Bayesian (VB) are used to solve the tree-based hierarchical models [8][9]. Turbo AMP [10] also well exploits tree sparsity for compressive sensing with an iterative approximate message

passing approach. However, none of them has conducted numerous experiments on MR images to validate their superiority.

In existing CS-MRI models, the linear combination of total variation and wavelet sparse regularization is very popular [3,12-15]. Classical conjugate gradient decent method is first used to solve this problem [3]. TVCMRI [12] and RecPF [13] use an operator-splitting method and a variable splitting method to solve this problem respectively. FCSA [14,15] decomposes the original problem into two easy subproblems and separately solves each of them with FISTA [16,17]. They are the state-of-the-art algorithms for CS-MRI, but none of them utilizes tree sparsity prior to enhance their performance.

In this paper, we propose a new model for CS-MRI, which combines wavelet sparsity, gradient sparsity and tree sparsity seamlessly. In tree structure modeling, we assign each pair of parent-child wavelet coefficients to one group, which forces them to be zeros or non-zeros simultaneously. This is an overlapping group problem and hard to be solved directly. A new variable is introduced to decompose this problem to three simpler subproblems. Then each of subproblems has closed form solution or can be solved efficiently by existing techniques. We conduct extensive experiments to compare the proposed algorithm with the state-of-the-art CS-MRI algorithms and several tree sparsity algorithms. The proposed algorithm always achieves the best results in terms of SNR and computational time.

Our contribution can be summarized as: (1) We introduce the wavelet tree sparsity to CS-MRI, and provide a convex formulation to model the tree-based structure combining with total variation and wavelet sparsity; (2) An efficient algorithm with fast convergence performance is proposed in this paper to solve this model. Each iteration only cost $\mathcal{O}(n \log n)$ time.(3) Numerous experiments have been conducted to compare the proposed algorithm with the state-of-the-art CS-MRI algorithms and several general tree-based algorithms or solvers. The results show that the proposed algorithm outperforms all others on real MR images.

## 2 Related work

### 2.1 Tree based compressive sensing

If a signal is sparse or can be sparsely represented, the necessary samples to reconstruct it can be significantly smaller than that needed in Nyquist-Shannon sampling theorem. Moreover, if we know some prior about the structure of the original signal, such as group or graph structure, the measurements can be further reduced [4,5]. Some previous algorithms have utilized the tree structure of wavelet coefficients to improve CS reconstruction [7-10].

OGL [7] is a convex approach to model the tree structure:

$$\hat{\theta} = \arg\min_{\theta}\{F(\theta) = \frac{1}{2}||b - A\Phi^T\theta||_2^2 + \lambda_g \sum_{g \in \mathcal{G}} ||\widetilde{\theta}_g||_2 + \frac{1}{2}\tau^2 \sum_{i=1}^{n}\sum_{j \in J_i}(\theta_i - \theta_i^j)^2\} \qquad (1)$$

where $\theta$ is a set of the wavelet coefficients. $A$ represents a partial Fourier transform for MR reconstruction problem and b is the measurements data. $\Phi^T$ denotes the inverse wavelet transform. $\mathcal{G}$ denotes the set of all parent-child groups and $g$ is one of such groups. $\widetilde{\theta}$ is an extended vector of $\theta$ with replicates and the last term is a penalty to force the replicates to be the same. When wavelet coefficients are recovered, they can be transformed to the recovered image by an inverse wavelet transform.This method well explores the tree structure assumption, but may be slow in general as following reasons: a) the parent-child relationship in their model is hard to maintain; b) it applies SpaRSA [11] to solve (1). Overall, their method can only achieve a convergence rate of $F(\theta^k) - F(\theta^*) \simeq \mathcal{O}(1/k)$ [16], where $k$ is the iteration number and $\theta^*$ is the optimal solution.

In statistical learning, AMP [10], MCMC [8], and VB [9] all solve (2) with probabilistic inference. In (2), $x$ is the original image to be reconstructed and $w$ is Gaussian white noise. In these approaches, graphical models are used to represent the wavelet tree structure and the distribution of each coefficient is decided by its parent's value.

$$y = Ax + w = A\Phi^T\theta + w \qquad (2)$$

## 2.2 Efficient MR image reconstruction algorithms

In existing CS-MRI algorithms, the linear combination of total variation and wavelet sparsity constrains has shown good property for MR images. Recent fastest algorithms all attempt to solve (3) in less computational time. $\alpha$ and $\beta$ are two positive parameters, and $\Phi$ denotes the wavelet transform. $\|x\|_{TV} = \sum_i \sum_j \sqrt{(\nabla_1 x_{ij})^2 + (\nabla_2 x_{ij})^2}$, where $\nabla_1$ and $\nabla_2$ denote the forward finite difference operators on the first and second coordinates. TVCMRI [12] and RecPF [13] use an operator-splitting method and a variable splitting method to solve this problem respectively. FCSA [14,15] decomposes this problem into 2 simpler problems and solves them with FISTA respectively. The convergence rate of FISTA is $\mathcal{O}(1/k^2)$. These approaches are very effective on real MR image reconstruction, but none of them utilizes the wavelet tree structure to get further enhancement.

$$\hat{x} = \arg\min_x \{\frac{1}{2}\|Ax - b\|_2^2 + \alpha\|x\|_{TV} + \beta\|\Phi x\|_1\} \tag{3}$$

## 2.3 Convex overlapped group sparsity solvers

SLEP [18] (Sparse Learning with Efficient Projections) has the package for tree structured group lasso (4). Its main function is to iteratively solve the tree structured denoising problem. When it comes to reconstruction problem, it applies FISTA to transfer the problem to denoising.

$$\hat{x} = \arg\min_x \{\frac{1}{2}\|Ax - b\|_2^2 + \beta\|\Phi x\|_{tree}\} \tag{4}$$

YALL1 [19] (Your Algorithms for L1) can solve the general overlapping group sparse problem efficiently. We put it in comparisons too. It first relaxes the constrained overlapping group minimization to unconstrained problem by Lagrangian method. Then the minimization of the $x$ and $z$ subproblems can be written as:

$$\hat{x} = \arg\min_{x,z} \{\frac{\beta_2}{2}\|Ax - b\|_2^2 + \lambda_1^T G\Phi x + \frac{\beta_1}{2}\|z - G\Phi x\|_2^2 - \lambda_2^T Ax + \sum_{i=1}^{s} w_i\|z_i\|_2\} \tag{5}$$

where $G$ indicates the grouping index with all its elements to be 1 or 0. $s$ is the total number of groups. $\lambda_1$, $\lambda_2$ are multipliers and $\beta_1$, $\beta_2$ are positive parameters.

## 3 Algorithm

Observations tell us that the wavelet coefficients of real MR images tend to be quadtree structured [20], although not strictly. Moreover they are generally sparse in wavelet and gradient domain. So we utilize all the sparse prior in our model. A new algorithm called *Wavelet Tree Sparsity MRI* (WaTMRI) is proposed to efficiently solve this model. Tree-based MRI problem can be formulated as follows:

$$\min_x \{F(x) = \frac{1}{2}\|Ax - b\|_2^2 + \alpha\|x\|_{TV} + \beta(\|\Phi x\|_1 + \sum_{g\in\mathcal{G}} \|\Phi x_g\|_2)\} \tag{6}$$

The total variation and $L1$ term in fact have complemented the tree structure assumption, which make our model more robust on real MR images. This is a main difference with previous tree structured algorithms or solvers. However, this problem can not be solved efficiently. We introduce

a variable $z$ to constrain $x$ with overlapping structure. Then the problem becomes non-overlapping convex optimization. Let $G\Phi x = z$, (6) can be rewritten as:

$$\min_{x,z}\{F(x) = \frac{1}{2}\|Ax - b\|_2^2 + \alpha\|x\|_{TV} + \beta(\|\Phi x\|_1 + \sum_{i=1}^{s}\|z_{g_i}\|_2) + \frac{\lambda}{2}\|z - G\Phi x\|_2^2\} \qquad (7)$$

For the $z$ subproblem, it has closed form solution by the group-wise soft thresholding. For the $x$ subproblem, we can combine the first and last quadratic penalty on the right side. Then the rest has the similar form with FCSA and can be solved efficiently with an iterative scheme.

## 3.1 Solution

As mentioned above, $z$-subproblem in (7) can be written as:

$$z_{g_i} = \arg\min_{z_{g_i}}\{\beta\|z_{g_i}\|_2 + \frac{\lambda}{2}\|z_{g_i} - (G\Phi x)_{g_i}\|_2^2\}, i = 1, 2, ..., s \qquad (8)$$

where $g_i$ is the i-th group and $s$ is number of total groups. It has a closed form solution by soft thresholding:

$$z_{g_i} = \max(\|r_i\|_2 - \frac{\beta}{\lambda}, 0)\frac{r_i}{\|r_i\|_2}, i = 1, 2, ..., s \qquad (9)$$

where $r_i = (G\Phi x)_{g_i}$.

For the $x$-subproblem,

$$x = \arg\min_x\{\frac{1}{2}\|Ax - b\|_2^2 + \alpha\|x\|_{TV} + \beta\|\Phi x\|_1 + \frac{\lambda}{2}\|z - G\Phi x\|_2^2\} \qquad (10)$$

Let $f(x) = \frac{1}{2}\|Ax - b\|_2^2 + \frac{\lambda}{2}\|z - G\Phi x\|_2^2$, which is a convex and smooth function with Lipschitz $L_f$, and $g_1(x) = \alpha\|x\|_{TV}$, $g_2(x) = \beta\|\Phi x\|_1$, which are convex but non-smooth functions. Then this $x$ problem can be solved efficiently by FCSA. For convenience, we denote (9) by $z = shrinkgroup(G\Phi x, \frac{\beta}{\lambda})$. Now, we can summarize our algorithm as in Algorithm 1:

---
**Algorithm 1** WaTMRI

**Input:** $\rho = 1/L_f, r^1 = x^0, t^1 = 1, \alpha, \beta, \lambda$
**for** $k = 1$ **to** $N$ **do**
   1)   $z = shrinkgroup(G\Phi x^{k-1}, \beta/\lambda)$
   2)   $x_g = r^k - \rho\nabla f(r^k)$
   3)   $x_1 = prox_\rho(2\alpha\|x\|_{TV})(x_g)$
   4)   $x_2 = prox_\rho(2\beta(\|\Phi x\|_1))(x_g)$
   5)   $x^k = (x_1 + x_2)/2$
   6)   $t^{k+1} = [1 + \sqrt{1 + 4(t^k)^2}]/2$
   7)   $r^{k+1} = x^k + \frac{t^k-1}{t^{k+1}}(x^k - x^{k-1})$
**end for**

---

where the proximal map is defined for any scaler $\rho > 0$:

$$prox_\rho(g)(x) := \arg\min_u\{g(u) + \frac{1}{2\rho}\|u - x\|^2\} \qquad (11)$$

and $\nabla f(r^k) = A^T(Ar^k - b) + \lambda\Phi^TG^T(G\Phi r^k - z)$ with $A^T$ denotes the inverse partial Fourier transform.

## 3.2 Algorithm analysis

Suppose $x$ represents an image with $n$ pixels and $z$ contains $n'$ elements. Although $G$ is a $n' \times n$ matrix, it is sparse with only $n'$ non-zero elements. So we can implement a multiplication by $G$ efficiently with $\mathcal{O}(n')$ time. Step 1 *shrinkgroup* takes $\mathcal{O}(n' + n \log n)$ time. The total cost of step 2 takes $\mathcal{O}(n \log n)$ time. Step 4 takes $\mathcal{O}(n \log n)$ when the fast wavelet transform is applied. Steps 3,5 all cost $\mathcal{O}(n)$. Note that $n' \leq 2n$ since we assign every parent-child coefficients to one group and leave every wavelet scaling in one group. So the total computation complexity for each iteration is $\mathcal{O}(n \log n)$, the same complexity as that in TVCMRI, RecPF and FCSA. We introduce the wavelet tree structure constrain in our model, without increasing the total computation complexity. The $x$-subproblem is accelerated by FISTA, and the whole algorithm shows a very fast convergence rate in the following experiments.

# 4 Experiments

## 4.1 Experiments set up

Numerous experiments have been conducted to show the superiority of the proposed algorithm on CS-MRI. In the MR imaging problem, $A$ is partial Fourier transform with $m$ rows and $n$ columns. We define the sampling ratio as $m/n$. The fewer measurements we samples, the less MR scanning time is need. So MR imaging is always interested in low sampling ratio cases. We follow the sampling strategy of previous works([12,14-15]), which randomly choose more Fourier coefficients from low frequency and less on high frequency. All measurements are mixed with $0.01$ Gaussian white noise. Signal-to-Noise Ratio (SNR) is used for result evaluation.

All experiments are on a laptop with 2.4GHz Intel core i5 2430M CPU. Matlab version is 7.8(2009a). We conduct experiments on four MR images : "Cardiac", "Brain", "Chest" and "Shoulder" ( Figure 1). We first compare our algorithm with the classical and fastest MR image reconstruction algorithms: CG[3], TVCMRI[12], RecPF[13], FCSA[14,15], and then with general tree based algorithms or solvers: AMP[10], VB[9], YALL1[19], SLEP[18]. For fair comparisons, all codes are downloaded from the authors' websites. We do not include MCMC[7] in experiements because it has slow execution speed and untractable convergence [9][10]. OGL[7] solves its model by SpaRSA [11] with only $\mathcal{O}(1/k)$ convergence rate, which can not be competitive with recent FISTA[16,17] algorithms with $\mathcal{O}(1/k^2)$ convergence rate. The authors have not published the code yet. So we do not include OGL for comparisons neither. We use the same setting $\alpha = 0.001$, $\beta = 0.035$ in previous works [12,14,15] for all convex models. $\lambda = 0.2 \times \beta$ is used for our model.

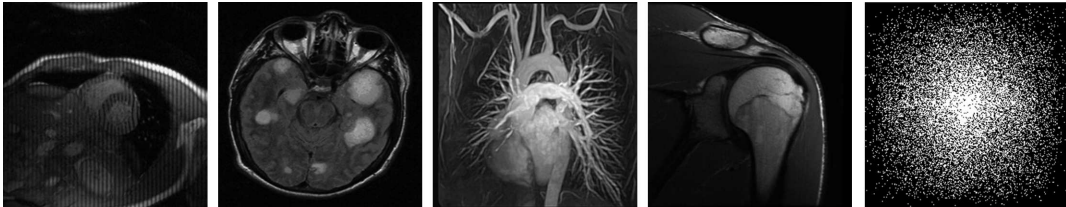

Figure 1: MR images: Cardiac; Brain; Chest; Shoulder and the sampling mask.

## 4.2 Comparisons with MR image reconstruction algorithms

We first compare our method with the state-of-the-art MR image reconstruction algorithms. For convenience, all test images are resized to $256 \times 256$. Figure 2 shows the performance comparison on "Brain" image. All algorithms terminate after 50 iterations. We decompose the wavelet coefficients to 4 levels here since more levels would increase the computation cost and less levels will weaken tree structure benefit. One could observe that the visual result recovered by the proposed algorithm is the closest to the original with only $20\%$ sampling ratio. Although tree structure definitely cost a little more time to solve, it always achieves the best performance in terms of SNR and CPU time. We have conducted experiments on other images and obtain similar results that the proposed algorithm always has the best performance in terms of SNR and CPU time. This result is reasonable because

we exploit wavelet tree structure in our model, which can reduce requirement for the number of measurements or increase the accuracy of the solution with the same measurements.

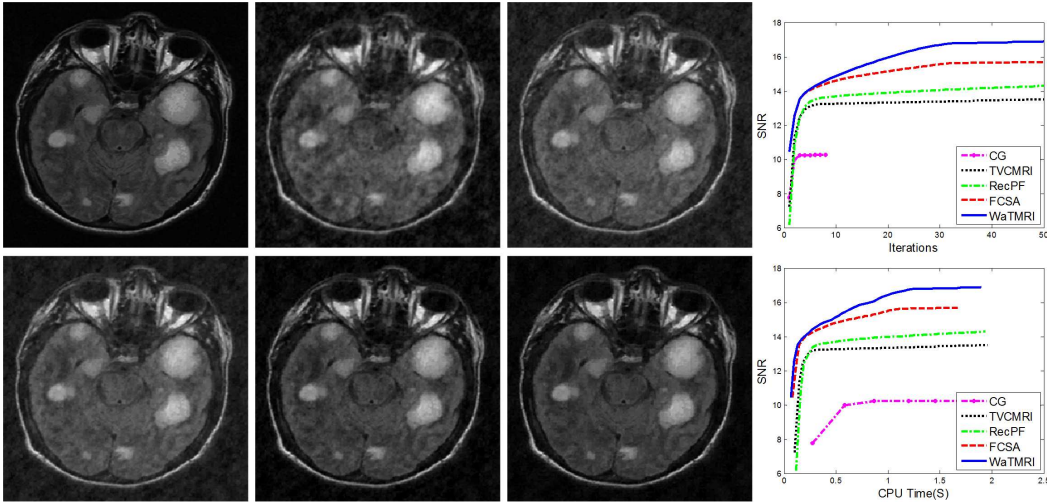

Figure 2: Brain image reconstruction with 20% sampling. Visual results from left to right, top to bottom are original image, images reconstructed by CG [3], TVCMRI [12], RecPF [13], FCSA [14,15], and the proposed algorithm. Their SNR are 10.26, 13.50, 14.29, 15.69 and 16.88. The right side shows the average SNR to iterations and SNR to CPU time.

## 4.3   Comparisons with general algorithms of tree structure

We also compare our algorithm with existing algorithms for tree sparsity with statistical inference and convex optimization. For statistical algorithms AMP [10] and VB [9], we use the default setting in their code. For SLEP [18], we set the same parameters $\alpha$ and $\beta$ as those in previous experiments. For YALL1 [19], we set both $\beta_1$ and $\beta_2$ equal to $\beta$. VB needs every column of $A$, which slows down the whole algorithm. Due to the higher space requirement and time complexity of VB, we resize all images to $128 \times 128$. The wavelet decomposition level is set as 3. Figure 3 shows the reconstruction results on "Brain" image with only 20% measurements. All algorithm terminates after 50 iterations. Due to high computational complexity of VB, we do not show the performance of VB in the right bottom panel. As AMP and VB can converge with only a small number of iterations and are much slower, we run them 10 iterations in all later experiments. The proposed algorithm always achieves the highest SNR to CPU time among all tree-based algorithms or solvers. These results are reasonable because none of the other algorithms uses the sparse prior of MR images in wavelet and gradient domain simultaneously.

Table 1: Comparisons of SNR (db) on four MR images

| Algorithms | Iterations | Cardiac | Brain | Chest | Shoulder |
|---|---|---|---|---|---|
| AMP [10] | 10 | 11.36±0.95 | 11.56±0.60 | 11.00±0.30 | 14.49±1.04 |
| VB [9] | 10 | 9.62±1.82 | 9.23±1.39 | 8.93±0.79 | 13.81±0.44 |
| SLEP [18] | 50 | 12.24±1.08 | 12.28±0.78 | 12.34±0.28 | 15.65±1.78 |
| YALL1 [19] | 50 | 9.56±0.13 | 7.73±0.15 | 7.76±0.56 | 13.14±0.22 |
| Proposed | 50 | **14.80± 0.51** | **14.11± 0.41** | **12.90± 0.13** | **18.93± 0.73** |

Table 1 and 2 show the results on four MR images. Although statistical algorithms are slow in general, they have the convenience without tuning parameters, as all parameters are learned from data. Fortunately, good parameters for MR image reconstruction are easy to tune in our model.

Except the proposed algorithm, all other algorithms have a strong assumption of the tree structure. However for real MR data, many images do not strictly follow this assumption. Due to this rea-

Table 2: Comparisons of execution time(sec) on four MR images

| Algorithms | Iterations | Cardiac | Brain | Chest | Shoulder |
|---|---|---|---|---|---|
| AMP [10] | 10 | 2.30±0.06 | 2.36±0.33 | 2.37±0.41 | 2.29±0.22 |
| VB [9] | 10 | 13.95±0.11 | 14.25±0.29 | 14.11±0.40 | 14.15±0.42 |
| SLEP [18] | 50 | 1.44±0.08 | 1.52±0.06 | 1.41±0.05 | 1.45±0.08 |
| YALL1 [19] | 50 | 1.02±0.04 | 1.04±0.01 | 0.98±0.04 | 1.00±0.02 |
| Proposed | 50 | 1.54±0.04 | 1.61±0.03 | 1.56±0.07 | 1.62±0.14 |

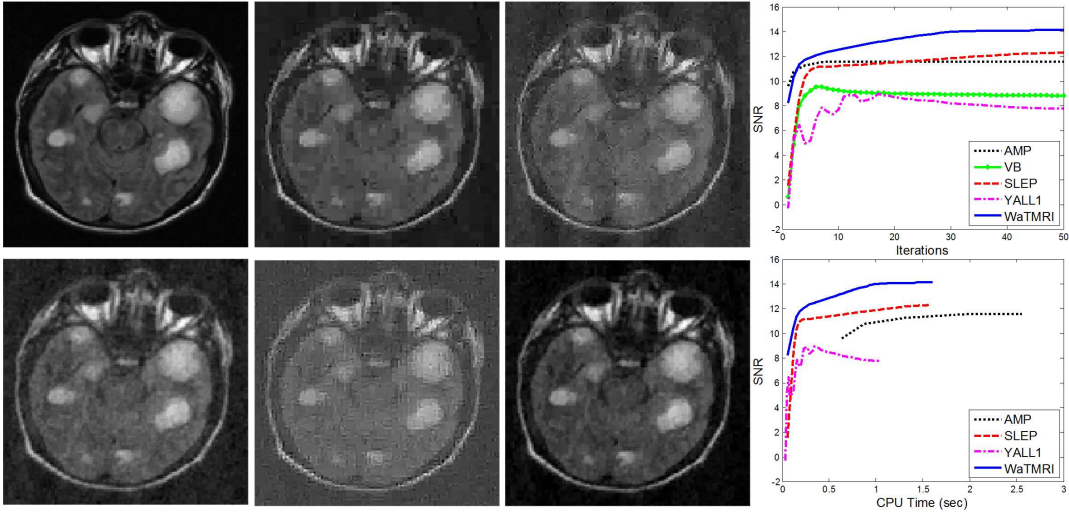

Figure 3: Brain image reconstruction with 20% sampling. Visual results from left to right, top to bottom are original image, images reconstructed by AMP [10], VB [9], SLEP [18], YALL1 [19], and the proposed algorithm. Their SNR are 11.56, 8.81, 12.28, 7.73 and 14.11. The right side shows the average SNR to iterations and to CPU time. Note that the right bottom panel only shows the first 10 iterations time of AMP.

son, these tree-based algorithms can not do their best on real MR images. To show the benefit of proposed model, we design another experiment on a toy MR image, which more strictly follow the tree structure assumption. First we set the wavelet coefficients who have the smallest 0.1% energy to zero. Then if a coefficient's parent or child is zero, we set it to be zero. Hence coefficients in the same group are both zeros or non-zeros. Figure 4 shows the original toy brain image and corresponding results of different algorithms. We found that all algorithms have improved a lot and the performance of all algorithms becomes much closer. From Figure 4 and 3, we could find that the proposed algorithm has great superiority on real MR image, because we combined TV and wavelet sparsity, which "soften" and complement the tree structure assumption for real MR data. Other tree based algorithms depend on "hard" tree structure only, which makes them hard to perform well on CS-MRI.

Finally, we show the results at different sampling ratios at Figure 5. For the same algorithm, the SNR of the solution tends to be higher when more measurements are used. On the same tested image, the order of performance tends to be the same. It coincides the conclusion in previous paper [14,15] that FCSA is better than TVCMRI and RecPF, and far better than classical method CG. Comparing all these experiments, the proposed algorithm always achieves the highest SNR than all other algorithms on real MR images.

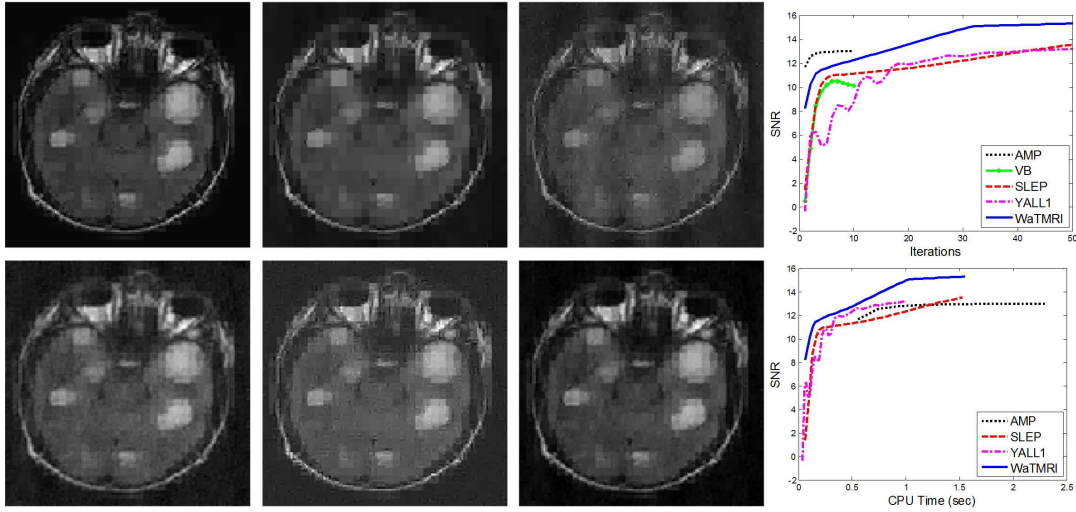

Figure 4: Toy image reconstruction with 20% sampling. Visual results from left to right, top to bottom are original image, images reconstructed by AMP [10], VB [9], SLEP [18], YALL1 [19], and the proposed algorithm. Their SNR are 12.99, 10.12, 13.53, 13.19 and 15.29. The right side shows the average SNR to iterations and to CPU time.

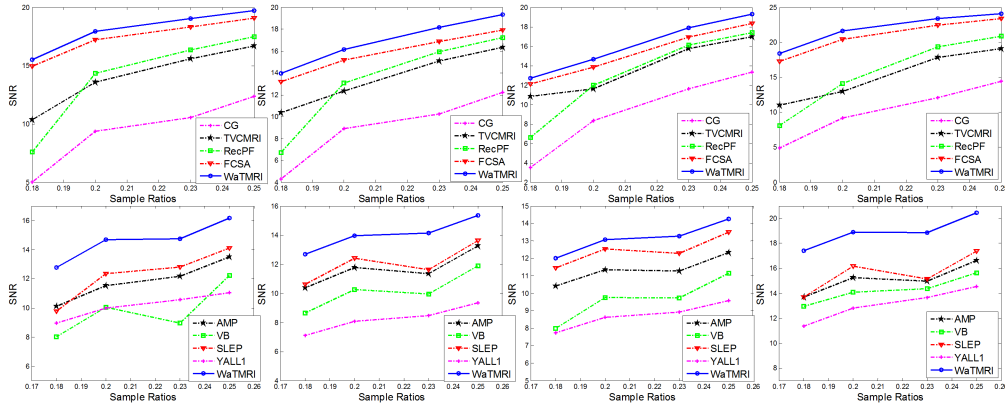

Figure 5: Average SNR with different sampling ratios on 4 MR images. All algorithms terminate after 50 iterations except AMP [10] and VB [9] terminate after 10 iterations. From left to right, results are on "Cardiac","Brain","Chest" and "Shoulder".

## 5   Conclusions

Real MR images not only tend to be tree structured sparse, but also are sparse in wavelet and gradient domain . In this paper, we consider all these priors in our model and all terms in this mode are convex. To solve this model, we decompose the original problem into three simpler ones and solve each of them very efficiently. Numerous experiments have been conducted to validate our method. All experiments demonstrate that the proposed algorithm outperforms the state-of-the-art ones in CS-MRI and general tree-based algorithms or solvers. Compared with the state-of-the-art algorithms in CS-MRI, the tree structure in our model can help reduce the required measurements, and leads to better performance. Compared with general tree sparsity algorithms, our algorithm can obtain more robust results on real MR data. Future work will be combining the proposed algorithm with the nonlocal total variation [22] for multi-contrast MRI [21].

## References

[1] Donoho, D. (2006) Compressed sensing. *IEEE Trans. on Information Theory* **52**(4):1289-1306.

[2] Candes, E., Romberg, J. & Tao, T. (2006) Robust uncertainty principles: Exact signal reconstruction from highly incomplete frequency information. *IEEE Trans. on Information Theory* **52**(2):489-509.

[3] Lustig, M., Donoho, D. & Pauly, J. (2007) Sparse MRI: The application of compressed sensing for rapid MR imaging. *Magnetic Resonance in Medicine* **58**(6):1182-1195.

[4] Huang, J., Zhang, T. & Metaxas, D. (2011) Learning With Structured Sparsity. *Journal of Machine Learning Research* **12**:3371-3412.

[5] Baraniuk, R.G., Cevher, V., Duarte, M.F. & Hegde, C. (2010) Model-based compressive sensing. *IEEE Trans. on Information Theory* **56**:1982-2001.

[6] Bach, F., Jenatton, R., Mairal, J. & Obozinski, G. (2012) Structured sparsity through convex optimization. Technical report, HAL 00621245-v2, *to appear in Statistical Science*.

[7] Rao, N., Nowak, R., Wright, S. & Kingsbury, N. (2011) Convex approaches to model wavelet sparsity patterns. In *IEEE International Conference On Image Processing, ICIP'11*

[8] He, L. & Carin, L. (2009) Exploiting Structure in Wavelet-Based Bayesian Compressive Sensing. *IEEE Trans. on Signal Processing* **57**(9):3488-3497.

[9] He, L., Chen, H. & Carin, L. (2010) Tree-Structured Compressive Sensing with Variational Bayesian Analysis. *IEEE Signal Processing Letters* **17**(3):233-236

[10] Som, S., Potter, L.C. & Schniter, P. (2010) Compressive Imaging using Approximate Message Passing and a Markov-Tree Prior. *In Proceedings of Asilomar Conference on Signals, Systems, and Computers*.

[11] Wright, S.J., Nowak, R.D. & Figueiredo, M.A.T. (2009) Sparse reconstruction by separable approximation. *IEEE Trans. on Signal Processing* **57**:2479-2493.

[12] Ma, S., Yin, W., Zhang, Y. & Chakraborty, A.(2008) An efficient algorithm for compressed MR imaging using total variation and wavelets. In *In Proc. of the IEEE Computer Society Conference on Computer Vision and Pattern Recognition*, CVPR'08.

[13] Yang, J., Zhang, Y. & Yin, W. (2010) A fast alternating direction method for tvl1-l2 signal reconstruction from partial fourier data. *IEEE Journal of Selected Topics in Signal Processing, Special Issue on Compressive Sensing* **4**(2):288-297.

[14] Huang, J., Zhang, S. & Metaxas, D. (2011) Efficient MR Image Reconstruction for Compressed MR Imaging. *Medical Image Analysis* **15**(5):670-679.

[15] Huang, J., Zhang, S. & Metaxas, D. (2010) Efficient MR Image Reconstruction for Compressed MR Imaging. *In Proc. of the 13th Annual International Conference on Medical Image Computing and Computer Assisted Intervention*, MICCAI'10.

[16] Beck, A. & Teboulle, M. (2009) A fast iterative shrinkage-thresholding algorithm for linear inverse problems. *SIAM Journal on Imaging Sciences* **2**(1):183-202.

[17] Beck, A. & Teboulle, M. (2009) Fast gradient-based algorithms for constrained total variation image denoising and deblurring problems. *IEEE Trans. on Image Processing* **18**(113):2419-2434

[18] Liu, J., Ji, S. & Ye, J. (2009) SLEP: Sparse Learning with Efficient Projections. Arizona State University. http://www.public.asu.edu/ jye02/Software/SLEP.

[19] Deng, W., Yin, W. & Zhang, Y. (2011) Group Sparse Optimization by Alternating Direction Method. Rice CAAM Report TR11-06.

[20] Manduca A., & Said A. (1996) Wavelet Compression of Medical Images with Set Partitioning in Hierarchical Trees. In *Proceedings of International Conference IEEE Engineering in Medicine and Biology Society, EMBS*.

[21] Huang, J., Chen, C. & Axel, L. (2012) Fast Multi-contrast MRI Reconstruction. *In Proc. of the 15th Annual International Conference on Medical Image Computing and Computer Assisted Intervention*, MICCAI'12.

[22] Huang, J., & Yang, F. (2012) Compressed Magnetic Resonace Imaging Based on Wavelet Sparsity and Nonlocal Total Variation. *IEEE International Symposium on Biomedical Imaging*, ISBI'12.

